# A Hybrid Neural Net System for State-of-the-Art Continuous Speech Recognition

**G. Zavaliagkos**
Northeastern University
Boston MA 02115

**Y. Zhao**
BBN Systems and Technologies
Cambridge, MA 02138

**R. Schwartz**
BBN Systems and Technologies
Cambridge, MA 02138

**J. Makhoul**
BBN Systems and Technologies
Cambridge, MA 02138

## Abstract

Untill recently, state-of-the-art, large-vocabulary, continuous speech recognition (CSR) has employed Hidden Markov Modeling (HMM) to model speech sounds. In an attempt to improve over HMM we developed a hybrid system that integrates HMM technology with neural networks. We present the concept of a "Segmental Neural Net" (SNN) for phonetic modeling in CSR. By taking into account all the frames of a phonetic segment simultaneously, the SNN overcomes the well-known conditional-independence limitation of HMMs. In several speaker-independent experiments with the DARPA Resource Management corpus, the hybrid system showed a consistent improvement in performance over the baseline HMM system.

## 1 INTRODUCTION

The current state of the art in continuous speech recognition (CSR) is based on the use of hidden Markov models (HMM) to model phonemes in context. Two main reasons for the popularity of HMMs are their high performance, in terms of recognition accuracy, and their computational efficiency However, the limitations of HMMs in modeling the speech signal have been known for some time. Two such limitations are (a) the conditional-independence assumption, which prevents a HMM from taking full advan-

tage of the correlation that exists among the frames of a phonetic segment, and (b) the awkwardness with which segmental features can be incorporated into HMM systems. We have developed the concept of Segmental Neural Nets (SNN) to overcome the two HMM limitations just mentioned for phonetic modeling in speech. A segmental neural net is a neural network that attempts to recognize a complete phonetic segment as a single unit, rather than a sequence of conditionally independent frames.

Neural nets are known to require a large amount of computation, especially for training. Also, there is no known efficient search technique for finding the best scoring segmentation with neural nets in continuous speech. Therefore, we have developed a hybrid SNN/HMM system that is designed to take full advantage of the good properties of both methods. The two methods are integrated through a novel use of the N-best (multiple hypotheses) paradigm developed in conjunction with the BYBLOS system at BBN [1].

## 2   SEGMENTAL NEURAL NET MODELING

There have been several recent approaches to the use of neural nets in CSR. The SNN differs from these approaches in that it attempts to recognize each phoneme by using all the frames in a phonetic segment simultaneously to perform the recognition. By looking at a whole phonetic segment at once, we are able to take advantage of the correlation that exists among frames of a phonetic segments, thus ameliorating the limitations of HMMs.

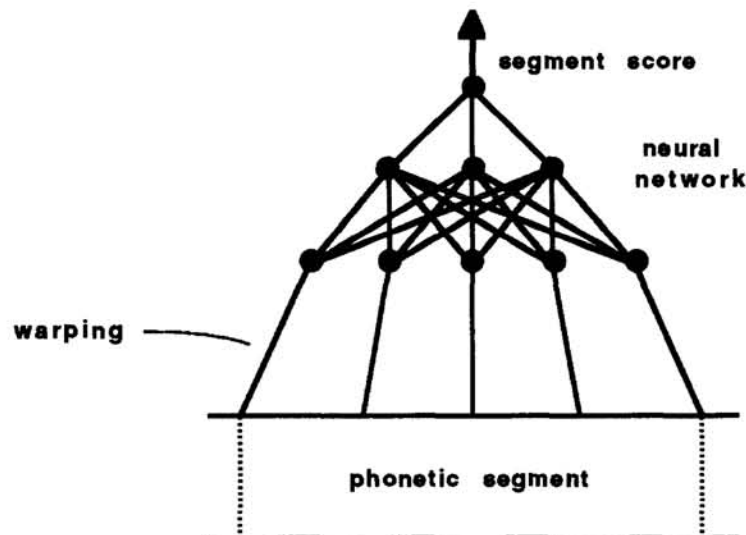

Figure 1: The SNN model samples the frames and produces a single segment score.

The structure of a typical SNN is shown in Figure 1. The input to the network is a fixed length representation of the speech segment. This input is scored by the network. If the network was trained to minimize a mean square error (MSE) or a relative entropy distortion measure, the output of the network will be an estimate of the posterior probability $P(C|x)$ of the phonetic class $C$ given the segment $x$ [2, 3]. This property of the SNN allows a natural extension to CSR: We segment the utterance into phonetic segments, and score each one of them seperately. The score of the utterance is simply the product of the scores of the individual segments.

The procedure described above requires the availability of some form of phonetic segmentation of the speech. We describe in Section 3 how we use the HMM to obtain likely candidate segmentations. Here, we shall assume that a phonetic segmentation has been made available and each segment is represented by a sequence of frames of speech features. The actual number of such frames in a phonetic segment is variable. However, for input to the neural network, we need a fixed length representation. Therefore, we have to convert the variable number of frames in each segment to a fixed number of frames. We have considered two approaches to cope with this problem: time sampling and Discrete Cosine Transform (DCT).

In the first approach, the requisite time warping is performed by a quasi-linear sampling of the feature vectors comprising the segment to a fixed number of frames (5 in our system). For example, in a 17-frame phonetic segment, we use frames 1, 5, 9, 13, and 17 as input to the SNN. The second approach uses the Discrete Cosine Transform (DCT). The DCT can be used to represent the frame sequence of a segment as follows. Consider the sequence of cepstral features across a segment as a time sequence and take its DCT. For an $m$ frame segment, this transform will result in a set of $m$ DCT coefficients for each feature. Truncate this sequence to its first few coefficients (the more coefficients, the more precise the representation). To keep the number of features the same as in the quasi-linear sampling, we use only five coefficients. If the input segment has less than five frames, we initially interpolate in time so that a five-point DCT is possible. Compared to the quasi-linear sampling, DCT has the advantage of using information from all input frames.

**Duration:** Because of the time warping function, the SNN score for a segment is independent of the duration of the segment. In order to provide duration information to the SNN, we constructed a simple durational model. For each phoneme, a histogram was made of segment durations in the training data. This histogram was then smoothed by convolving with a triangular window, and probabilities falling below a floor level were reset to that level. The duration score was multiplied by the neural net score to give an overall segment score.

## 3   THE N-BEST RESCORING PARADIGM

Our hybrid system is based on the N-best rescoring paradigm [1], which allows us to design and test the SNN with little regard to the usual problem of searching for the segmentation when dealing with a large vocabulary speech recognition system.

Figure 2 illustrates the hybrid system. Each utterance is decoded using the BBN BYBLOS system [4]. The decoding is done in two steps: First the N-best recognition is performed, producing a list of the candidate N best-scoring sentence hypotheses. In this stage, a relatively simple HMM is used for computation purposes. The length of the N-best list is chosen to be long enough to almost always include the correct answer. The second step is the HMM rescoring, where a more sophisticated HMM is used. The recognition process may stop at this stage, selecting the top scoring utterance of the list (HMM 1-best output).

To incorporate the SNN in the N-best paradigm, we use the HMM system to generate a segmentation for each N-best hypothesis, and the SNN to generate a score for the hypothesis using the HMM segmentation. The N-best list may be reordered based on

SNN scores alone. In this case the recognition process stops by selecting the top scoring utterance of the rescored list (NN 1-best output).

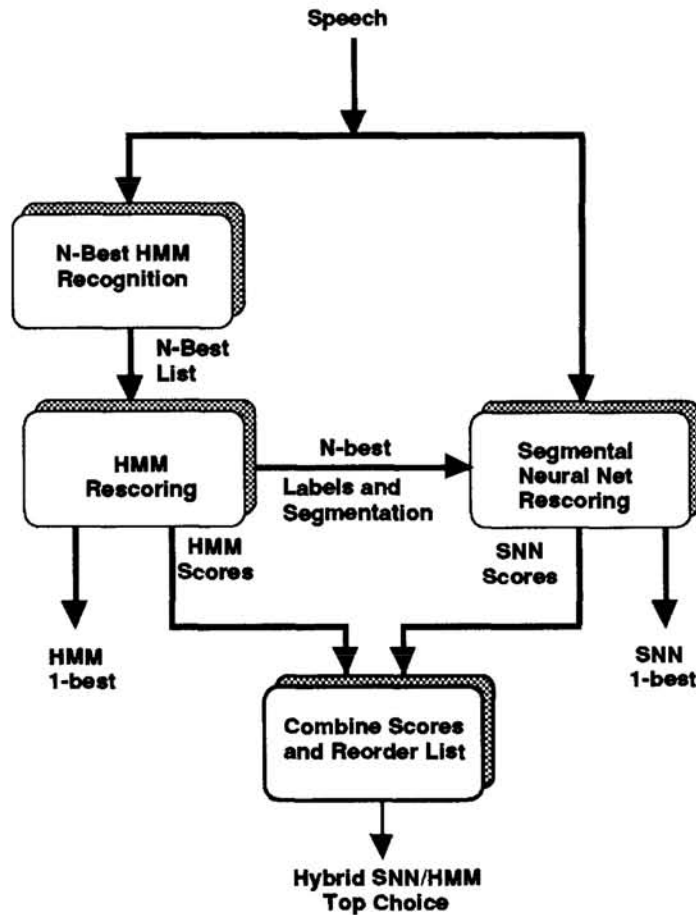

Figure 2: Schematic diagram of the hybrid SNN/HMM system

The last stage in the hybrid system is to combine several scores for each hypothesis, such as SNN score, HMM score, grammar score, and the hypothesized number of words and phonemes. (The number of words and phonemes are included because they serve the same purpose as word and phoneme insertion penalties in a HMM CSR system.) We form a composite score by taking a linear combination of the individual scores. The linear combination is determined by selecting the weights that give the best performance over a development test set. These weights can be chosen automatically [5]. After we have rescored the N-Best list, we can reorder it according to the new composite scores. If the CSR system is required to output just a single hypothesis, the highest scoring hypothesis is chosen (hybrid SNN/HMM top choice in Figure 2).

## 4    SNN TRAINING

The training of the phonetic SNNs is done in two steps. In the first training step, we segment all of the training utterances into phonetic segments using the HMM models and

the utterance transcriptions. Each segment then serves as a positive example of the SNN output corresponding to the phonetic label of the segment and as a negative example for all the other phonetic SNN outputs (we are using a total of 53 phonetic outputs). We call this training method *1-best training*.

The SNN is trained using the log-error distortion measure [6], which is an extension of the relative entropy measure to an $M$-class problem. To ensure that the outputs are in fact probabilities, we use a sigmoidal nonlinearity to restrict their range in [0, 1] and an output normalization layer to make them sum to one. The models are initialized by removing the sigmoids and using the MSE measure. Then we reinstate the sigmoids and proceed with four iterations of a quasi-Newton [7] error minimization algorithm. For the adopted error measure, when the neural net non-linearity is the usual sigmoid function, there exists a unique minimum for single-layer nets [6].

The 1-best training described has one drawback: the training does not cover all the cases that the network will be required to encounter in the N-best rescoring paradigm. With 1-best training, given the correct segmentation, we train the network to discriminate between correct and incorrect labeling. However, the network will also be used to score N-best hypotheses with incorrect segmentation. Therefore, it is important to train based on the N-best lists in what we call N-best training. During N-best training, we produce the N-best lists for all of the training sentences, and we then train positively with all the correct hypotheses and negatively on the "misrecognized" parts of the incorrect hypothesis.

### 4.1   Context Modelling

Some of the largest gains in accuracy for HMM CSR systems have been obtained with the use of context (i.e., phonetic identity of neighboring segments). Consequently, we implemented a version of the SNN that provided a simple model of left-context. In addition to the SNN previously described, which only models a segment's phonetic identity and makes no reference to context, we trained 53 additional left-context networks. Each of these 53 networks were identical in structure to the context-independent SNN. In the recognition process, the segment score is obtained by combining the output of the context-independent SNN with the corresponding output of the SNN that models the left-context of the segment. This combination is a weighted average of the two network values, where the weights are determined by the number of occurrences of the phoneme in the training data and the number of times the phoneme has its present context in the training data.

### 4.1.1   Regularization Techniques for Context Models

During neural net training of context models, a decrease of the distortion on the training set often causes an increase of the distortion on the test set. This problem is called overtraining, and it typically occurs when the number of training samples is on the order of the number of the model parameters. Regularization provides a class of smoothing techniques to ameliorate the overtraining problem. Instead of minimizing the distortion measure alone, we are minimizing the following objective function:

$$Distortion\_measure + \frac{\lambda_1}{N_d^{\eta_1}}\|\vec{W}\|^2 + \frac{\lambda_2}{N_d^{\eta_2}}\|\vec{W} - \vec{W}_0\|^2 \qquad (1)$$

where $\vec{W}_0$ is the set of weights corresponding to the context-independent model, $N_d$ is the number of data points, and $\lambda_1, \lambda_2, \eta_1, \eta_2$ are smoothing parameters. The first regularization term is used to control the excursion of the weights in general and the other to control the degree to which the context-dependent model is allowed to deviate from the corresponding context-independent model (to achieve this first we initialize the context-dependent models with the context-independent model). In our initial experiments, we used values of $\lambda_1 = \lambda_2 = 1.0$, $\eta_1 = 1$, $\eta_2 = 2$.

When there are very few training data for a particular context model, the regularization terms in (1) prevail, constraining the model parameters to remain close to their initial estimates. The regularization term is gradually turned off with the presence of more data. What we accomplish in this way is an automatic mechanism that controls overtraining.

## 4.2 Elliptical Basis Functions

Our efforts to use multi-layer structures has been rather unsuccessful so far. The best improvement we got was a mere 5% reduction in error rate over the single-layer performance, but with a 10-fold increase in both number of parameters and computation time. We suspect that our training is getting trapped in bad local minima. Due to the above considerations, we considered an alternative multi-layer structure, the Elliptical Basis Function (EBF) network. EBFs are natural extensions of Radial Basis Functions, where a full covariance matrix is introduced in the basis functions. As many researchers have suggested, EBF networks provide modelling capabilities that are as powerful as multi-layer perceptrons. An advantage of EBF is that there exist well established techniques for estimating the elliptical basis layer. As a consequence, the problem of training an EBF network can be reduced to a one-layer problem, i.e., training the second layer only.

Our approach with EBF is to initialize them with Maximum Likelihood (ML). ML training allows us to use very detailed context models, such as triphones. The next step, which is not yet implemented, is to either proceed with discriminative NN training, or use a nonlinearity at the outout layer and treat the second layer as a single-layer feedforward model, or both.

# 5   EXPERIMENTAL CONDITIONS AND RESULTS

Experiments to test the performance of the hybrid system were performed on the speaker-independent (SI) portion of the DARPA 1000-word Resource Management speech corpus. The training set consisted of utterances from 109 speakers, 2830 utterances from male speakers and 1160 utterances from female speakers. We have tested our system with 5 different test sets. The Feb '89 set was used as a cross-validation set for the SNN system. Feb '89 and Oct '89 were used as development sets whenever the weights for the combination of two or more models were to be estimated. Feb '91 and the two Sep '92 sets were used as independent test sets.

Both the NN and the HMM systems had 3 separate models made from male, female, and combined data. During recognition all 3 models were used to score the utterances, and the recognition answer was decided by a 3-way gender selection: For each utterance, the model that produced the highest score was selected. The HMM used was the February '91 version of the BBN BYBLOS system.

In the experiments, we used SNNs with 53 outputs, each representing one of the phonemes in our system. The SNN was used to rescore N-best lists of length N = 20. The input to the net is a fixed number of frames of speech features (5 frames in our system). The features in each 10-ms frame consist of 16 scalar values: power, power difference, and 14 mel-warped cepstral coefficients. For the EBF, the differences of the cepstral parameters were used also.

Table 1: SNN development on February '89 test set

|   |   | Word Error (%) |
|---|---|---|
|   | Original SSN (MSE) | 13.7 |
| + | Log-Error Criterion | 11.6 |
| + | N-Best training | 9.0 |
| + | Left Context | 7.4 |
| + | Regularization | 6.6 |
| + | word,phoneme penalties | 5.7 |
|   | EBF | 4.9 |

Table 1 shows the word error rates at the various stages of development. All the experiments mentioned below used the Feb '89 test set. The original 1-layer SNN was trained using the 1-best training algorithm and the MSE criterion, and gave a word error rate of 13.7%. The incorporation of the duration and the adoption of the log-error training criterion both resulted in some improvement, bringing the error rate down to 11.6%. With N-best training the error rate dropped to 9.0%; adding left context models reduced the word error rate down to 7.4%. When the the context models were trained with the regularization criterion the error rate dropped to 6.6%. All of the above results were obtained using the mean NN score (NN score divided by the number of segments). When we used word and phone penalties, the performance was even better, a 5.7% word error rate. For the same conditions, the performance for the EBF system was 4.9% word error rate. We should mention that the implementation of training with regularization was not complete at the time the hybrid system was tested on the September 92 test, so we will exclude it from the NN results presented below.

The final hybrid system included the HMM, the SNN and EBF models, and Table 2 summarizes its performance (in this table, NN stands for the combination of SNN and EBF). We notice that with the exception of of the Sep '92 test sets the word error of the HMM was roughly around 3.5%(3.8, 3.7 and 3.4%). For the same test sets, the NN had a word error slightly higher than 4.0%, and the hybrid NN/HMM system a word error rate of 2.7%. We are very happy to see the performance of our neural net approaching the performance of the HMM. It is also worthwhile to mention that the performance of the hybrid system for Feb '89, Oct '89 and Feb '91 is the best performance reported so far for these sets.

Special mention has to be made for the Sep '92 test sets. These test sets proved to be radically different than the previous released RM tests, resulting in almost a doubling of the HMM word error rate. The deterioration in performance of the hybrid system was bigger, and the improvement due to the hybrid system was less than 10% (compared to an improvement of $\approx$ 25% for the other 3 sets). We have all been baffled by these unexpected results, and although we are continuously looking for an explanation of this

|  | Word Error % | | | |
| System | Feb '89 | Oct '89 | Feb '91 | Sep '92 |
| --- | --- | --- | --- | --- |
| HMM | 3.7 | 3.8 | 3.4 | 6.0 |
| NN | 4.0 | 4.2 | 4.1 | 7.4 |
| NN+HMM | 2.7 | 2.7 | 2.7 | 5.5 |

Table 2: Hybrid Neural Net/HMM system.

strange behaviour our efforts have not yet been successful.

## 6   CONCLUSIONS

We have presented the Segmental Neural Net as a method for phonetic modeling in large vocabulary CSR systems and have demonstrated that, when combined with a conventional HMM, the SNN gives a significant improvement over the performance of a state-of-the-art HMM CSR system. The hybrid system is based on the N-best rescoring paradigm which, by providing the HMM segmentation, drastically reduces the computation for our segmental models and provides a simple way of combining the best aspects of two systems. The improvements achieved from the use of a hybrid system vary from less than 10% to about 25% reduction in word error rate, depending on the test set used.

## References

[1] R. Schwartz and S. Austin, "A Comparison of Several Approximate Algorithms for Finding Multiple (N-Best) Sentence Hypotheses," *IEEE Int. Conf. Acoustics, Speech and Signal Processing*, Toronto, Canada, May 1991, pp. 701-704.

[2] A. Barron, "Statistical properties of artificial neural networks," *IEEE Conf. Decision and Control*, Tampa, FL, pp. 280-285, 1989.

[3] H. Gish, "A probabilistic approach to the understanding and training of neural network classifiers," *IEEE Int. Conf. Acoust., Speech, Signal Processing*, April 1990.

[4] M. Bates et. all, "The BBN/HARC Spoken Language Understanding System" *IEEE Int. Conf. Acoust., Speech,Signal Processing*, Apr 1992, Minneapolis, MI, Apr. 1993

[5] M. Ostendorf et. all, "Integration of Diverse Recognition Methodologies Through Reevaluation of N-Best Sentence Hypotheses," *Proc. DARPA Speech and Natural Language Workshop*, Pacific Grove, CA, Morgan Kaufmann Publishers, February 1991.

[6] A. El-Jaroudi and J. Makhoul, "A New Error Criterion for Posterior Probability Estimation with Neural Nets," *International Joint Conference on Neural Networks*, San Diego, CA, June 1990, Vol III, pp. 185-192.

[7] D. Luenberger, *Linear and Nonlinear Programming*, Addison-Wesley, Massachusetts, 1984.

[8] R. Schwartz et. all, "Improved Hidden Markov Modeling of Phonemes for Continuous Speech Recognition," *IEEE Int. Conf. Acoustics, Speech and Signal Processing*, San Diego, CA, March 1984, pp. 35.6.1–35.6.4.
